# Subgrouping Reduces Complexity and Speeds Up Learning in Recurrent Networks

David Zipser
Department of Cognitive Science
University of California, San Diego
La Jolla, CA 92093

## 1 INTRODUCTION

Recurrent nets are more powerful than feedforward nets because they allow simulation of dynamical systems. Everything from sine wave generators through computers to the brain are potential candidates, but to use recurrent nets to emulate dynamical systems we need learning algorithms to program them. Here I describe a new twist on an old algorithm for recurrent nets and compare it to its predecessors.

## 2 BPTT

In the beginning there was BACKPROPAGATION THROUGH TIME (BPTT) which was described by Rumelhart, Williams, and Hinton (1986). The idea is to add a copy of the whole recurrent net to the top of a growing feedforward network on each update cycle. Backpropagating through this stack corrects for past mistakes by adding up all the weight changes from past times. A difficulty with this method is that the feedforward net gets very big. The obvious solution is to truncate it at a fixed number of copies by killing an old copy every time a new copy is added. The truncated-BPTT algorithm is illustrated in Figure 1. It works well, more about this later.

## 3 RTRL

It turns out that it is not necessary to keep an ever growing stack of copies of the recurrent net as BPTT does. A fixed number of parameters can record all of past time. This is done in the REAL TIME RECURRENT LEARNING (RTRL) algorithm of Williams and Zipser (1989). The derivation is given elsewhere (Rumelhart, Hinton, & Williams, 1986), but a

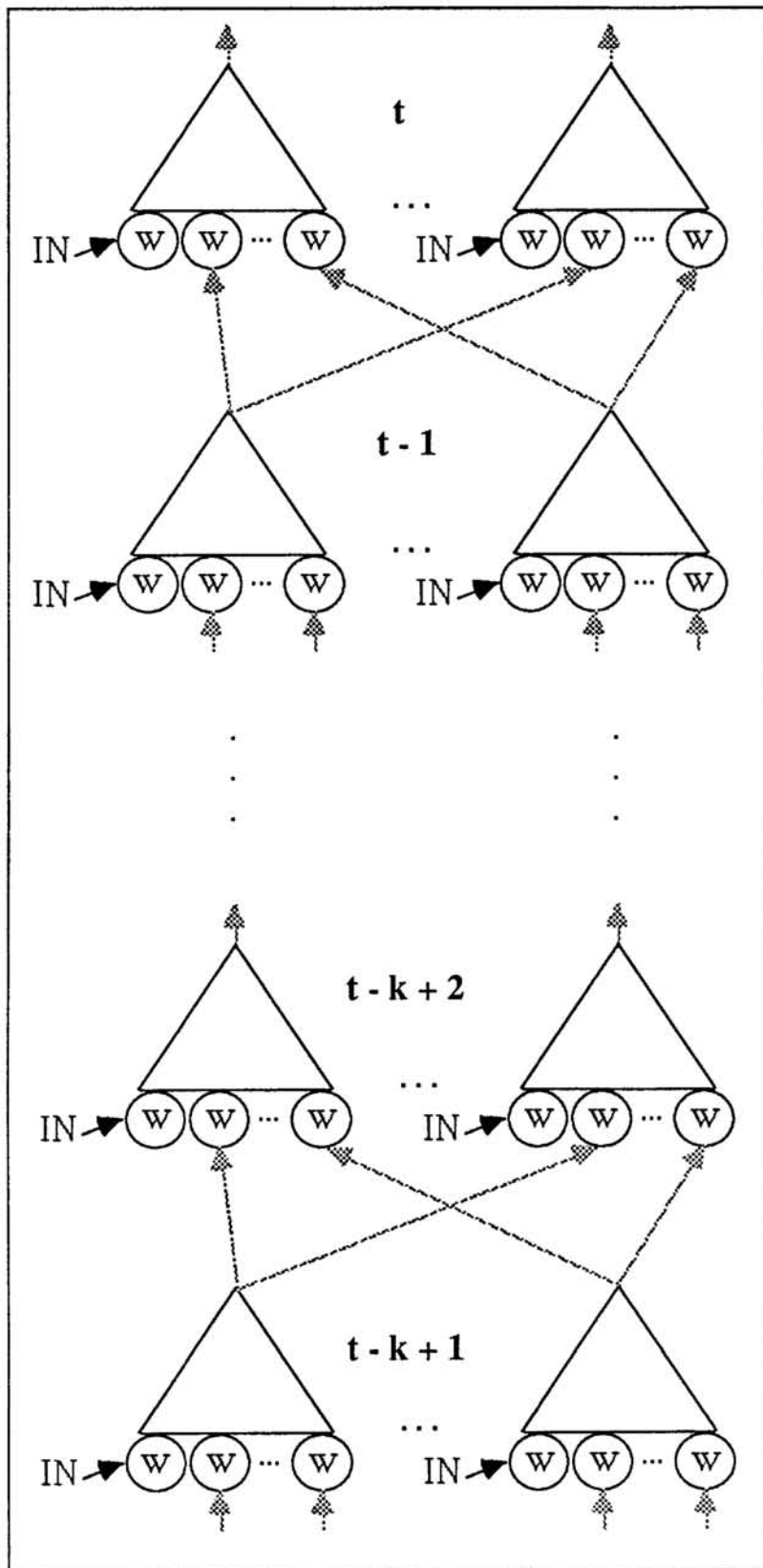

**Figure 1:** BPTT.

simple rational comes from the fact that error backpropagation is linear, which makes it possible to collapse the whole feedforward stack of BPTT into a few fixed size data structures. The biggest and most time consuming to update of these is the matrix of $p$ values whose update rule is

$$p_{ij}^{k}(t+1) = f'(s_k(t))\left[\sum_{l \in U} w_{kl}\, p_{ij}^{l}(t) + \delta_{ik}\, z_j(t)\right]$$

$$i \in U,\ j \in U \cup I,\ k \in U$$

where $z_k(t)$ represents the value of a signal, either an input or recurrent; the sets of subscriptss are defined so that if $z_k$ is an input then $k \in l$ and if $z_k$ is a signal from a recurrently connected unit then $k \in U$, $s_k$ are *net* values; $d_{ik}$ is the Kronecker delta; and $w_{kl}$ is the recurrent weight matrix. For a network with $n$ units and $w$ weights there are $nw$ of these $p$ values, and it takes $O(wn^2)$ operations to update them. As $n$ gets big this gets very big and is computationally unpleasant. This unpleasantness is cured to some degree by the new variant of RTRL described below.

## 4 SUBGROUPED RTRL

The value of $n$ in the factor $wn^2$, which causes all the trouble for RTRL, can be reduced by viewing a recurrent network as consisting of a set of subnetworks all connected together. A fully recurrent network with $n$ units and $m$ inputs can be divided into $g$ fully recurrent subnets, each with $n/g$ units (assuming $g$ is a factor of $n$). Each unit in a subnet will receive as input the original $m$ inputs and the activities of the $n - n/g$ units in the other subnets. The effect of subgrouping is to reduce the number of $p$ values per weight to $n/g$ and the number of operations to update the $p$ to $O(wn^2/g^2)$. If $g$ is increased in proportion to $n$, which keeps the size of the sub-nets constant, $n^2/g^2$ is a constant and the complexity is reduced to $O(w)$. If all this is confusing try Figure 2.

## 5 TESTING THESE ALGORITHMS

To see if the subgrouped algorithm works, I compared its performance to RTRL and BPTT on the problem of training a Turing machine to balance parentheses. The network "sees" the same tape as the Turing machine, and is trained to produce the same outputs. A fully recurrent network with 12 units was the smallest that learned this task. All three algorithms learned the task in about the same number of learning cycles. RTRL and subgrouped RTRL succeeded 50%, and BPTT succeeded 80% of the time. Subgrouped RTRL was 10 times faster than RTRL, whereas BPTT was 28 times faster.

### References

Rumelhart, D. E., Hinton, G. E., & Williams, R. J. (1986). Learning internal representations by error propagation. In D. E. Rumelhart, J. L. McClelland, & the PDP Research Group (Eds.), *Parallel distributed processing: Explorations in the microstructure of cognition. Vol. 1. Foundationa.* Cambridge, MA: MIT Press.

Williams, R. J., & Zipser, D. (1989). A learning algorithm for continually running fully recurrent neural networks. *Neural Computation, 1*, 270-280.

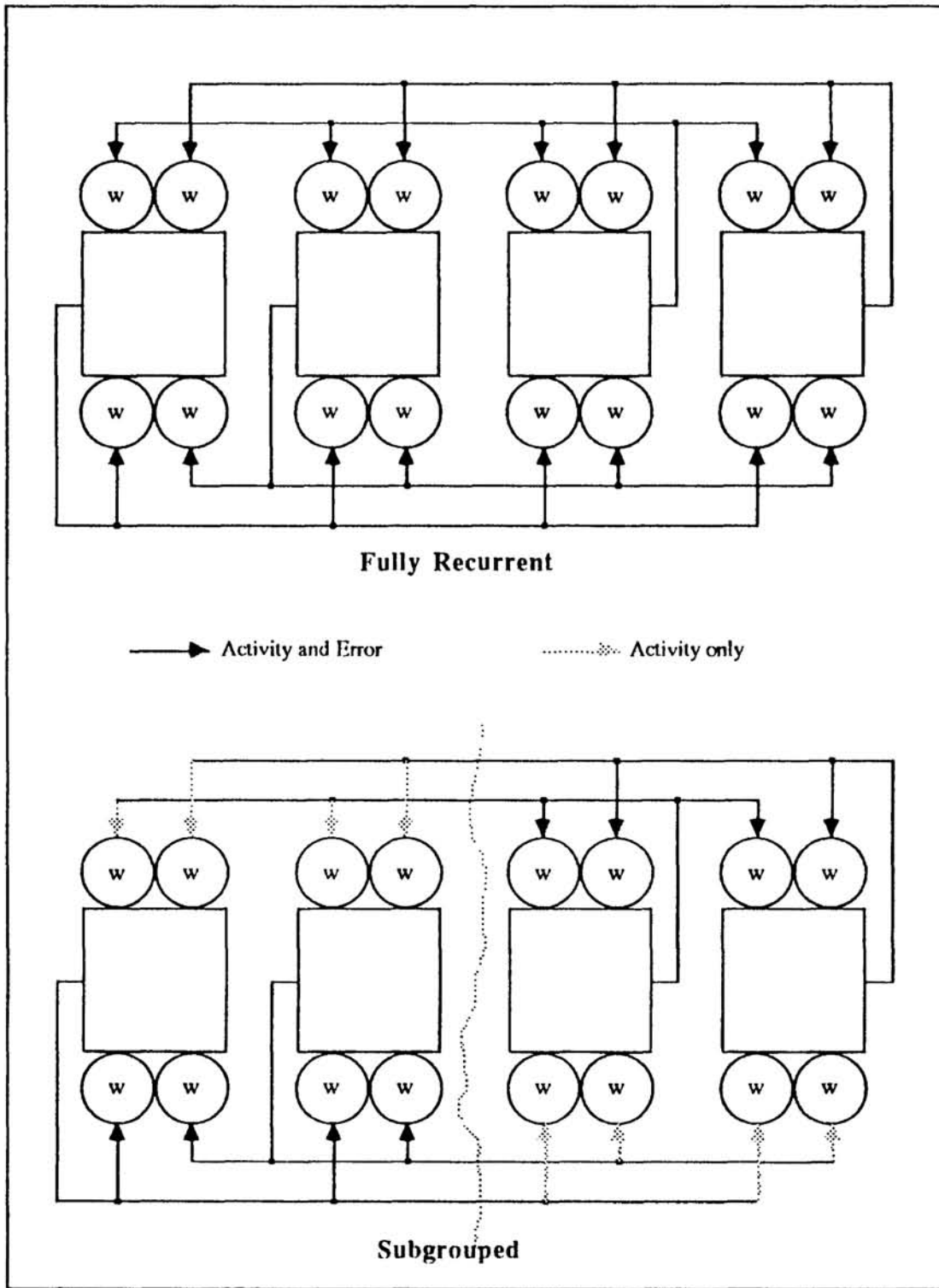

Figure 2: Subgrouped-RTRL